# Object discovery and identification

**Charles Kemp & Alan Jern**
Department of Psychology
Carnegie Mellon University
{ckemp,ajern}@cmu.edu

**Fei Xu**
Department of Psychology
University of California, Berkeley
fei_xu@berkeley.edu

## Abstract

Humans are typically able to infer how many objects their environment contains and to recognize when the same object is encountered twice. We present a simple statistical model that helps to explain these abilities and evaluate it in three behavioral experiments. Our first experiment suggests that humans rely on prior knowledge when deciding whether an object token has been previously encountered. Our second and third experiments suggest that humans can infer how many objects they have seen and can learn about categories and their properties even when they are uncertain about which tokens are instances of the same object.

From an early age, humans and other animals [1] appear to organize the flux of experience into a series of encounters with discrete and persisting objects. Consider, for example, a young child who grows up in a home with two dogs. At a relatively early age the child will solve the problem of *object discovery* and will realize that her encounters with dogs correspond to views of two individuals rather than one or three. The child will also solve the problem of *identification*, and will be able to reliably identify an individual (e.g. Fido) each time it is encountered.

This paper presents a Bayesian approach that helps to explain both object discovery and identification. Bayesian models are appealing in part because they help to explain how inferences are guided by prior knowledge. Imagine, for example, that you see some photographs taken by your friends Alice and Bob. The first shot shows Alice sitting next to a large statue and eating a sandwich, and the second is similar but features Bob rather than Alice. The statues in each photograph look identical, and probably you will conclude that the two photographs are representations of the same statue. The sandwiches in the photographs also look identical, but probably you will conclude that the photographs show different sandwiches. The prior knowledge that contributes to these inferences appears rather complex, but we will explore some much simpler cases where prior knowledge guides identification.

A second advantage of Bayesian models is that they help to explain how learners cope with uncertainty. In some cases a learner may solve the problem of object discovery but should maintain uncertainty when faced with identification problems. For example, I may be quite certain that I have met eight different individuals at a dinner party, even if I am unable to distinguish between two guests who are identical twins. In other cases a learner may need to reason about several related problems even if there is no definitive solution to any one of them. Consider, for example, a young child who must simultaneously discover which objects her world contains (e.g. Mother, Father, Fido, and Rex) and organize them into categories (e.g. people and dogs). Many accounts of categorization seem to implicitly assume that the problem of identification must be solved before categorization can begin, but we will see that a probabilistic approach can address both problems simultaneously.

Identification and object discovery have been discussed by researchers from several disciplines, including psychology [2, 3, 4, 5, 6], machine learning [7, 8], statistics [9], and philosophy [10]. Many machine learning approaches can handle identity uncertainty, or uncertainty about whether two tokens correspond to the same object. Some approaches such such as BLOG [8] are able in addition to handle problems where the number of objects is not specified in advance. We propose

that some of these approaches can help to explain human learning, and this paper uses a simple BLOG-style approach [8] to account for human inferences.

There are several existing psychological models of identification, and the work of Shepard [11], Nosofsky [3] and colleagues is probably the most prominent. Models in this tradition usually focus on problems where the set of objects is specified in advance and where identity uncertainty arises as a result of perceptual noise. In contrast, we focus on problems where the number of objects must be inferred and where identity uncertainty arises from partial observability rather than noise. A separate psychological tradition focuses on problems where the number of objects is not fixed in advance. Developmental psychologists, for example, have used displays where only one object token is visible at any time to explore whether young infants can infer how many different objects have been observed in total [4]. Our work emphasizes some of the same themes as this developmental research, but we go beyond previous work in this area by presenting and evaluating a computational approach to object identification and discovery.

The problem of deciding how many objects have been observed is sometimes called *individuation* [12] but here we treat individuation as a special case of object discovery. Note, however, that object discovery can also refer to cases where learners infer the existence of objects that have never been observed. *Unobserved-object discovery* has received relatively little attention in the psychological literature, but is addressed by statistical models including including species-sampling models [9] and capture-recapture models [13]. Simple statistical models of this kind will not address some of the most compelling examples of unobserved-object discovery, such as the discovery of the planet Neptune, or the ability to infer the existence of a hidden object by following another person's gaze [14]. We will show, however, that a simple statistical approach helps to explain how humans infer the existence of objects that they have never seen.

## 1 A probabilistic account of object discovery and identification

Object discovery and identification may depend on many kinds of observations and may be supported by many kinds of prior knowledge. This paper considers a very simple setting where these problems can be explored. Suppose that an agent is learning about a world that contains $n_w$ white balls and $n - n_w$ gray balls. Let $f(o_i)$ indicate the color of ball $o_i$, where each ball is white ($f(o_i) = 1$) or gray ($f(o_i) = 0$). An agent learns about the world by observing a sequence of object tokens. Suppose that label $l(j)$ is a unique identifier of token $j$—in other words, suppose that the $j$th token is a token of object $o_{l(j)}$. Suppose also that the $j$th token is observed to have feature value $g(j)$. Note the difference between $f$ and $g$: $f$ is a vector that specifies the color of the $n$ balls in the world, and $g$ is a vector that specifies the color of the object tokens observed thus far.

We define a probability distribution over token sequences by assuming that a world is sampled from a prior $P(n, n_w)$ and that tokens are sampled from this world. The full generative model is:

$$P(n) \quad \propto \begin{cases} \frac{1}{n} & \text{if } n \leq 1000 \\ 0 & \text{otherwise} \end{cases} \tag{1}$$

$$n_w \mid n \sim \text{Uniform}(0, n) \tag{2}$$

$$l(j) \mid n \sim \text{Uniform}(1, n) \tag{3}$$

$$g(j) \quad = f(o_{l(j)}) \tag{4}$$

A prior often used for inferences about a population of unknown size is the scale-invariant Jeffreys prior $P(n) = \frac{1}{n}$ [15]. We follow this standard approach here but truncate at $n = 1000$. Choosing some upper bound is convenient when implementing the model, and has the advantage of producing a prior that is proper (note that the Jeffreys prior is improper). Equation 2 indicates that the number of white balls $n_w$ is sampled from a discrete uniform distribution. Equation 3 indicates that each token is generated by sampling one of the $n$ balls in the world uniformly at random, and Equation 4 indicates that the color of each token is observed without noise.

The generative assumptions just described can be used to define a probabilistic approach to object discovery and identification. Suppose that the observations available to a learner consist of a fully-observed feature vector $g$ and a partially-observed label vector $l_{obs}$. Object discovery and identification can be addressed by using the posterior distribution $P(l|g, l_{obs})$ to make inferences about the number of distinct objects observed and about the identity of each token. Computing the posterior distribution $P(n|g, l_{obs})$ allows the learner to make inferences about the total number of objects

in the world. In some cases, the learner may solve the problem of unobserved-object discovery by realizing that the world contains more objects than she has observed thus far.

The next sections explore the idea that the inferences made by humans correspond approximately to the inferences of this ideal learner. Since the ideal learner allows for the possible existence of objects that have not yet been observed, we refer to our model as the *open world* model. Although we make no claim about the psychological mechanisms that might allow humans to approximate the predictions of the ideal learner, in practice we need some method for computing the predictions of our model. Since the domains we consider are relatively small, all results in this paper were computed by enumerating and summing over the complete set of possible worlds.

## 2   Experiment 1: Prior knowledge and identification

The introduction described a scenario (the statue and sandwiches example) where prior knowledge appears to guide identification. Our first experiment explores a very simple instance of this idea. We consider a setting where participants observe balls that are sampled with replacement from an urn. In one condition, participants sample the same ball from the urn on four consecutive occasions and are asked to predict whether the token observed on the fifth draw is the same ball that they saw on the first draw. In a second condition participants are asked exactly the same question about the fifth token but sample four different balls on the first four draws. We expect that these different patterns of data will shape the prior beliefs that participants bring to the identification problem involving the fifth token, and that participants in the first condition will be substantially more likely to identify the fifth token as a ball that they have seen before.

Although we consider an abstract setting involving balls and urns the problem we explore has some real-world counterparts. Suppose, for example, that a colleague wears the same tie to four formal dinners. Based on this evidence you might be able to estimate the total number of ties that he owns, and might guess that he is less likely to wear a new tie to the next dinner than a colleague who wore different ties to the first four dinners.

**Method.** 12 adults participated for course credit. Participants interacted with a computer interface that displayed an urn, a robotic arm and a beam of UV light. The arm randomly sampled balls from the urn, and participants were told that each ball had a unique serial number that was visible only under UV light. After some balls were sampled, the robotic arm moved them under the UV light and revealed their serial numbers before returning them to the urn. Other balls were returned directly to the urn without having their serial numbers revealed. The serial numbers were alphanumeric strings such as "QXR182"—note that these serial numbers provide no information about the total number of objects, and that our setting is therefore different from the Jeffreys tramcar problem [15].

The experiment included five within-participant conditions shown in Figure 1. The observations for each condition can be summarized by a string that indicates the number of tokens and the serial numbers of some but perhaps not all tokens. The ①①①①① condition in Figure 1a is a case where the same ball (without loss of generality, we call it ball 1) is drawn from the urn on five consecutive occasions. The ①②③④⑤ condition in Figure 1b is a case where five different balls are drawn from the urn. The ①◯◯◯◯ condition in Figure 1d is a case where five draws are made, but only the serial number of the first ball is revealed. Within any of the five conditions, all of the balls had the same color (white or gray), but different colors were used across different conditions. For simplicity, all draws in Figure 1 are shown as white balls.

On the second and all subsequent draws, participants were asked two questions about any token that was subsequently identified. They first indicated whether the token was likely to be the same as the ball they observed on the first draw (the ball labeled ① in Figure 1). They then indicated whether the token was likely to be a ball that they had never seen before. Both responses were provided on a scale from 1 (very unlikely) to 7 (very likely). At the end of each condition, participants were asked to estimate the total number of balls in the urn. Twelve options were provided ranging from "exactly 1" to "exactly 12," and a thirteenth option was labeled "more than 12." Responses to each option were again provided on a seven point scale.

**Model predictions and results.** The comparisons of primary interest involve the identification questions in conditions 1a and 1b. In condition 1a the open world model infers that the total number of balls is probably low, and becomes increasingly confident that each new token is the same as the

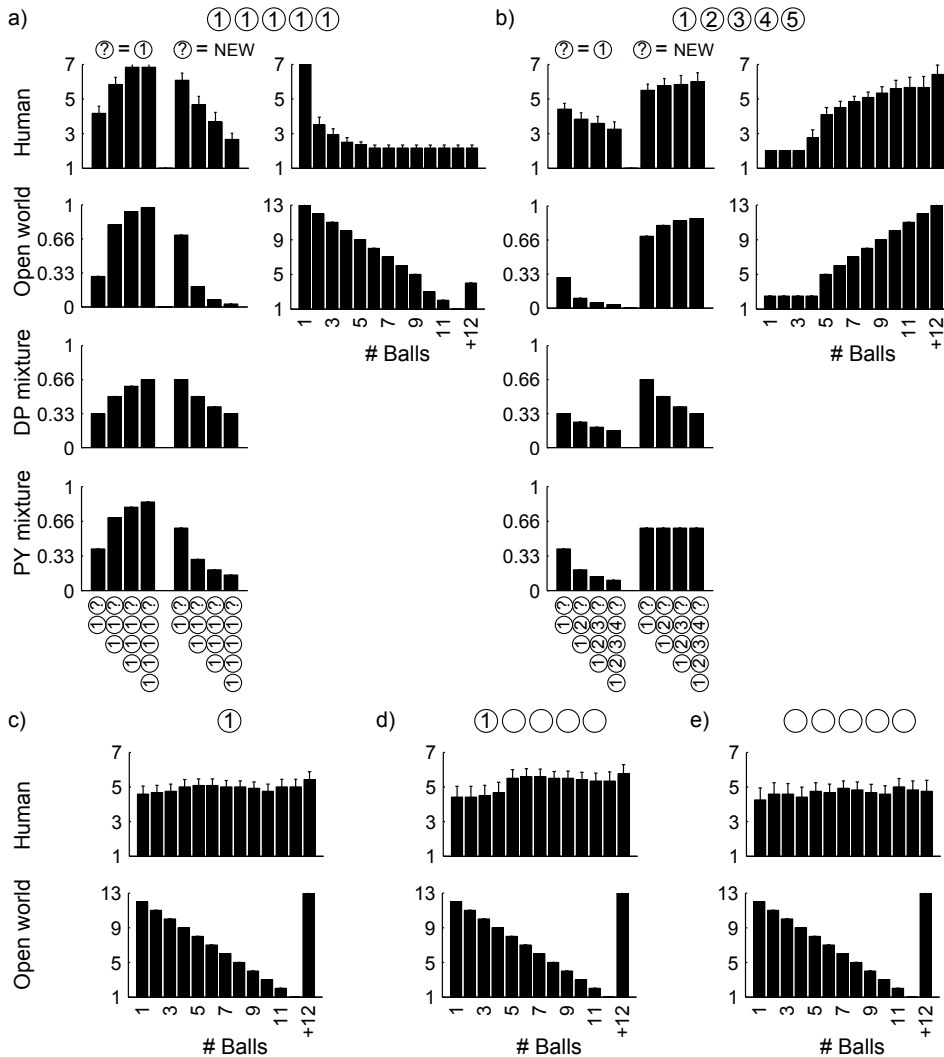

Figure 1: Model predictions and results for the five conditions in experiment 1. The left columns in (a) and (b) show inferences about the identification questions. In each plot, the first group of bars shows predictions about the probability that each new token is the same ball as the first ball drawn from the urn. The second group of bars shows the probability that each new token is a ball that has never been seen before. The right columns in (a) and (b) and the plots in (c) through (e) show inferences about the total number of balls in each urn. All human responses are shown on the 1-7 scale used for the experiment. Model predictions are shown as probabilities (identification questions) or ranks (population size questions).

first object observed. In condition 1b the model infers that the number of balls is probably high, and becomes increasingly confident that each new token is probably a new ball.

The rightmost charts in Figures 1a and 1b show inferences about the total number of balls and confirm that humans expect the number of balls to be low in condition 1a and high in condition 1b. Note that participants in condition 1b have solved the problem of unobserved-object discovery and inferred the existence of objects that they have never seen. The leftmost charts in 1a and 1b show responses to the identification questions, and the final bar in each group of four shows predictions about the fifth token sampled. As predicted by the model, participants in 1a become increasingly confident that each new token is the same object as the first token, but participants in 1b become increasingly confident that each new token is a new object. The increase in responses to the new ball questions in Figure 1b is replicated in conditions 2d and 2e of Experiment 2, and therefore appears to be reliable.

The third and fourth rows of Figures 1a and 1b show the predictions of two alternative models that are intuitively appealing but that fail to account for our results. The first is the Dirichlet Process (DP) mixture model, which was proposed by Anderson [16] as an account of human categorization. Unlike most psychological models of categorization, the DP mixture model reserves some probability mass for outcomes that have not yet been observed. The model incorporates a prior distribution over partitions—in most applications of the model these partitions organize objects into categories, but Anderson suggests that the model can also be used to organize object tokens into classes that correspond to individual objects. The DP mixture model successfully predicts that the ball ① questions will receive higher ratings in 1a than 1b, but predicts that responses to the new ball question will be identical across these two conditions. According to this model, the probability that a new token corresponds to a new object is $\frac{\theta}{m+\theta}$ where $\theta$ is a hyperparameter and $m$ is the number of tokens observed thus far. Note that this probability is the same regardless of the identities of the $m$ tokens previously observed.

The Pitman Yor (PY) mixture model in the fourth row is a generalization of the DP mixture model that uses a prior over partitions defined by two hyperparameters [17]. According to this model, the probability that a new token corresponds to a new object is $\frac{\theta+k\alpha}{m+\theta}$, where $\theta$ and $\alpha$ are hyperparameters and $k$ is the number of distinct objects observed so far. The flexibility offered by a second hyperparameter allows the model to predict a difference in responses to the new ball questions across the two conditions, but the model does not account for the increasing pattern observed in condition 1b. Most settings of $\theta$ and $\alpha$ predict that the responses to the new ball questions will decrease in condition 1b. A non-generic setting of these hyperparameters with $\theta = 0$ can generate the flat predictions in Figure 1, but no setting of the hyperparameters predicts the increase in the human responses. Although the PY and DP models both make predictions about the identification questions, neither model can predict the total number of balls in the urn. Both models assume that the population of balls is countably infinite, which does not seem appropriate for the tasks we consider.

Figures 1c through 1d show results for three control conditions. Like condition 1a, 1c and 1d are cases where exactly one serial number is observed. Like conditions 1a and 1b, 1d and 1e are cases where exactly five tokens are observed. None of these control conditions produces results similar to conditions 1a and 1b, suggesting that methods which simply count the number of tokens or serial numbers will not account for our results.

In each of the final three conditions our model predicts that the posterior distribution on the number of balls $n$ should decay as $n$ increases. This prediction is not consistent with our data, since most participants assigned equal ratings to all 13 options, including "exactly 12 balls" and "more than 12 balls." The flat responses in Figures 1c through 1e appear to indicate a generic desire to express uncertainty, and suggest that our ideal learner model accounts for human responses only after several informative observations have been made.

## 3   Experiment 2: Object discovery and identity uncertainty

Our second experiment focuses on object discovery rather than identification. We consider cases where learners make inferences about the number of objects they have seen and the total number of objects in the urn even though there is substantial uncertainty about the identities of many of the tokens observed. Our probabilistic model predicts that observations of unidentified tokens can influence inferences about the total number of objects, and our second experiment tests this prediction.

**Method.** 12 adults participated for course credit. The same participants took part in Experiments 1 and 2, and Experiment 2 was always completed after Experiment 1. Participants interacted with the same computer interface in both conditions, and the seven conditions in Experiment 2 are shown in Figure 2. Note that each condition now includes one or more gray tokens. In 2a, for example, there are four gray tokens and none of these tokens is identified. All tokens were sampled with replacement, and the condition labels in Figure 2 summarize the complete set of tokens presented in each condition. Within each condition the tokens were presented in a pseudo-random order—in 2a, for example, the gray and white tokens were interspersed with each other.

**Model predictions and results.** The cases of most interest are the inferences about the total number of balls in conditions 2a and 2c. In both conditions participants observe exactly four white tokens and all four tokens are revealed to be the same ball. The gray tokens in each condition are never identified, but the number of these tokens varies across the conditions. Even though the identities

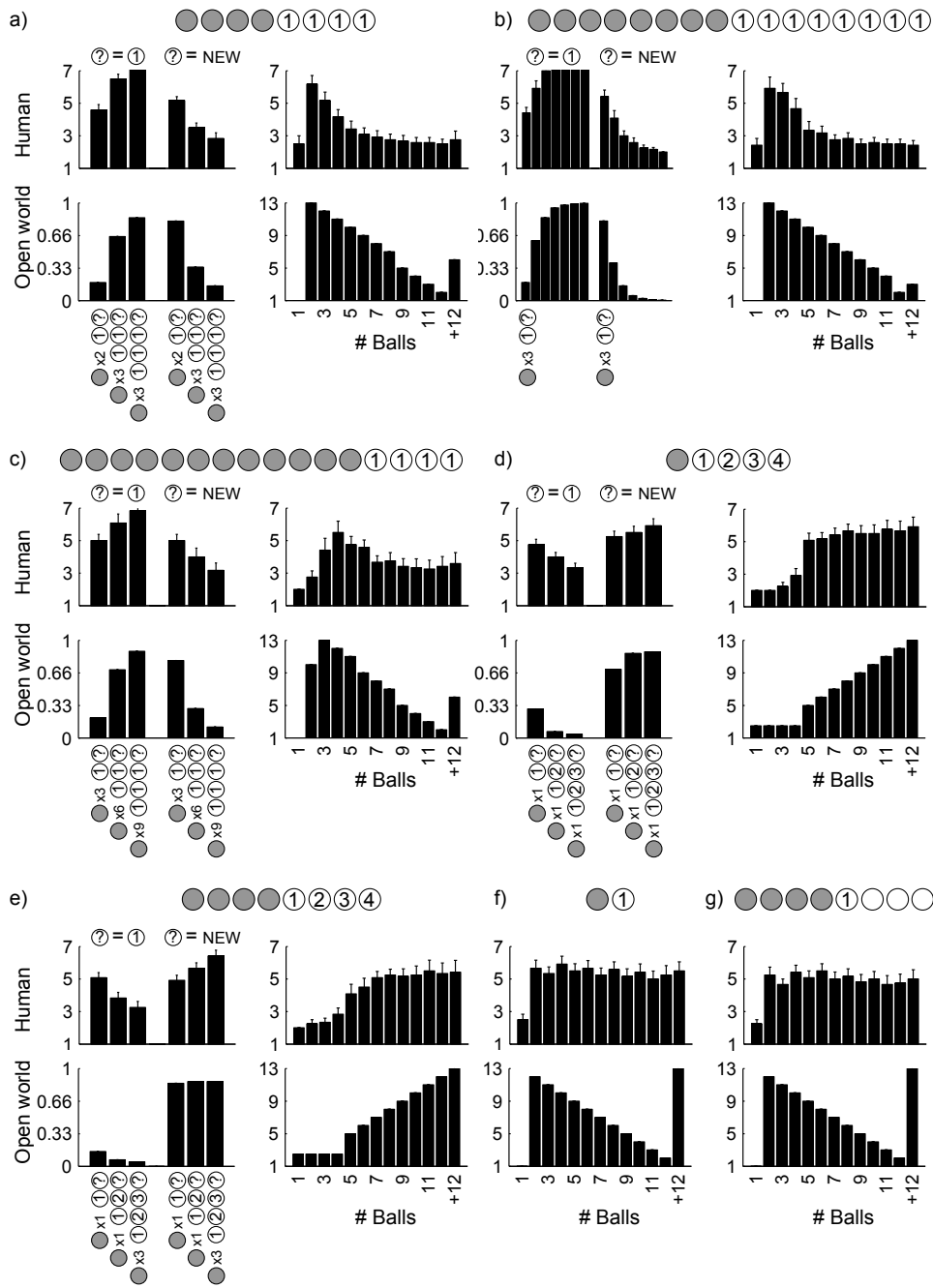

Figure 2: Model predictions and results for the seven conditions in Experiment 2. The left columns in (a) through (e) show inferences about the identification questions, and the remaining plots show inferences about the total number of balls in each urn.

of the gray tokens are never revealed, the open world model can use these observations to guide its inference about the total number of balls. In 2a, the proportions of white tokens and gray tokens are equal and there appears to be only one white ball, suggesting that the total number of balls is around two. In 2c grey tokens are now three times more common, suggesting that the total number of balls is larger than two. As predicted, the human responses in Figure 2 show that the peak of the distribution in 2a shifts to the right in 2c. Note, however, that the model does not accurately predict the precise location of the peak in 2c.

Some of the remaining conditions in Figure 2 serve as controls for the comparison between 2a and 2c. Conditions 2a and 2c differ in the total number of tokens observed, but condition 2b shows that

this difference is not the critical factor. The number of tokens observed is the same across 2b and 2c, yet the inference in 2b is more similar to the inference in 2a than in 2c. Conditions 2a and 2c also differ in the proportion of white tokens observed, but conditions 2f and 2g show that this difference is not sufficient to explain our results. The proportion of white tokens observed is the same across conditions 2a, 2f, and 2g, yet only 2a provides strong evidence that the total number of balls is low. The human inferences for 2f and 2g show the hint of an alternating pattern consistent with the inference that the total number of balls in the urn is even. Only 2 out of 12 participants generated this pattern, however, and the majority of responses are near uniform. Finally, conditions 2d and 2e replicate our finding from Experiment 1 that the identity labels play an important role. The only difference between 2a and 2e is that the four labels are distinct in the latter case, and this single difference produces a predictable divergence in human inferences about the total number of balls.

## 4    Experiment 3: Categorization and identity uncertainty

Experiment 2 suggested that people make robust inferences about the existence and number of unobserved objects in the presence of identity uncertainty. Our final experiment explores categorization in the presence of identity uncertainty. We consider an extreme case where participants make inferences about the variability of a category even though the tokens of that category have never been identified.

**Method.** The experiment included two between subject conditions, and 20 adults were recruited for each condition. Participants were asked to reason about a category including eggs of a given species, where eggs in the same category might vary in size. The interface used in Experiments 1 and 2 was adapted so that the urn now contained two kinds of objects: notepads and eggs. Participants were told that each notepad had a unique color and a unique label written on the front. The UV light played no role in the experiment and was removed from the interface: notepads could be identified by visual inspection, and identifying labels for the eggs were never shown.

In both conditions participants observed a sequence of 16 tokens sampled from the urn. Half of the tokens were notepads and the others were eggs, and all egg tokens were identical in size. Whenever an egg was sampled, participants were told that this egg was a Kwiba egg. At the end of the condition, participants were shown a set of 11 eggs that varied in size and asked to rate the probability that each one was a Kwiba egg. Participants then made inferences about the total number of eggs and the total number of notepads in the urn.

The two conditions were intended to lead to different inferences about the total number of eggs in the urn. In the 4 egg condition, all items (notepad and eggs) were sampled with replacement. The 8 notepad tokens included two tokens of each of 4 notepads, suggesting that the total number of notepads was 4. Since the proportion of egg tokens and notepad tokens was equal, we expected participants to infer that the total number of eggs was roughly four. In the 1 egg condition, four notepads were observed in total, but the first three were sampled without replacement and never returned to the urn. The final notepad and the egg tokens were always sampled with replacement. After the first three notepads had been removed from the urn, the remaining notepad was sampled about half of the time. We therefore expected participants to infer that the urn probably contained a single notepad and a single egg by the end of the experiment, and that all of the eggs they had observed were tokens of a single object.

**Model.** We can simultaneously address identification and categorization by combining the open world model with a Gaussian model of categorization. Suppose that the members of a given category (e.g. Kwiba eggs) vary along a single continuous dimension (e.g. size). We assume that the egg sizes are distributed according to a Gaussian with known mean and unknown variance $\sigma^2$. For convenience, we assume that the mean is zero (i.e. we measure size with respect to the average) and use the standard inverse-gamma prior on the variance: $p(\sigma^2) \propto (\sigma^2)^{-(\alpha+1)} e^{-\frac{\beta}{\sigma^2}}$. Since we are interested only in qualitative predictions of the model, the precise values of the hyperparameters are not very important. To generate the results shown in Figure 3 we set $\alpha = 0.5$ and $\beta = 2$.

Before observing any eggs, the marginal distribution on sizes is $p(x) = \int p(x|\sigma^2)p(\sigma^2)d\sigma^2$. Suppose now that we observe $m$ random samples from the category and that each one has size zero. If $m$ is large then these observations provide strong evidence that the variance $\sigma^2$ is small, and the posterior distribution $p(x|m)$ will be tightly peaked around zero. If $m$, is small, however, then the posterior distribution will be broader.

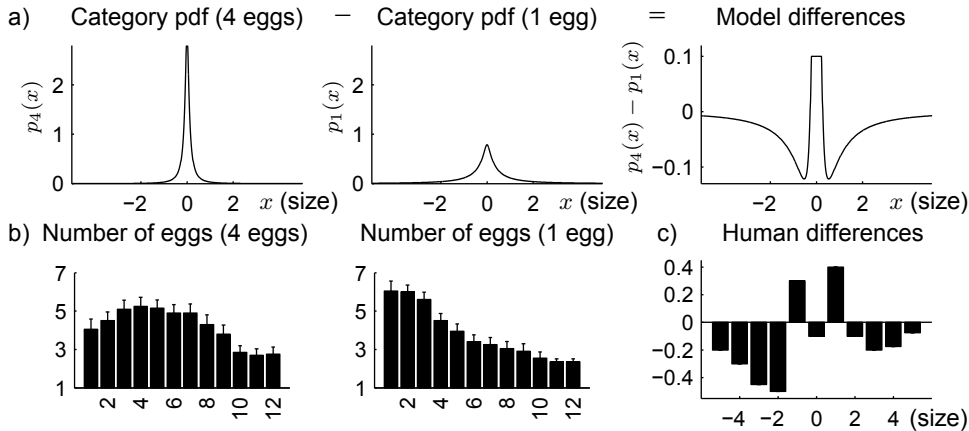

Figure 3: (a) Model predictions for Experiment 3. The first two panels show the size distributions inferred for the two conditions, and the final panel shows the difference of these distributions. The difference curve for the model rises to a peak of around 1.6 but has been truncated at 0.1. (b) Human inferences about the total number of eggs in the urn. As predicted, participants in the 4 egg condition believe that the urn contains more eggs. (c) The difference of the size distributions generated by participants in each condition. The central peak is absent but otherwise the curve is qualitatively similar to the model prediction.

The categorization model described so far is entirely standard, but note that our experiment considers a case where $T$, the observed stream of object tokens, is not sufficient to determine $m$, the number of distinct objects observed. We therefore use the open world model to generate a posterior distribution over $m$, and compute a marginal distribution over size by integrating out both $m$ and $\sigma^2$: $p(x|T) = \int p(x|\sigma^2)p(\sigma^2|m)p(m|T)d\sigma^2 dm$. Figure 3a shows predictions of this "open world + Gaussian" model for the two conditions in our experiment. Note that the difference between the curves for the two conditions has the characteristic Mexican-hat shape produced by a difference of Gaussians.

**Results.** Inferences about the total number of eggs suggested that our manipulation succeeded. Figure 3b indicates that participants in the 4 egg condition believed that they had seen more eggs than participants in the 1 egg condition. Participants in both conditions generated a size distribution for the category of Kwiba eggs, and the difference of these distributions is shown in Figure 3c. Although the magnitude of the differences is small, the shape of the difference curve is consistent with the model predictions. The $x = 0$ bar is the only case that diverges from the expected Mexican hat shape, and this result is probably due to a ceiling effect—80% of participants in both conditions chose the maximum possible rating for the egg with mean size (size zero), leaving little opportunity for a difference between conditions to emerge. To support the qualitative result in Figure 3c we computed the variance of the curve generated by each individual participant and tested the hypothesis that the variances were greater in the 1 egg condition than in the 4 egg condition. A Mann-Whitney test indicated that this difference was marginally significant ($p < 0.1$, one-sided).

## 5    Conclusion

Parsing the world into stable and recurring objects is arguably our most basic cognitive achievement [2, 10]. This paper described a simple model of object discovery and identification and evaluated it in three behavioral experiments. Our first experiment confirmed that people rely on prior knowledge when solving identification problems. Our second and third experiments explored problems where the identities of many object tokens were never revealed. Despite the resulting uncertainty, we found that participants in these experiments were able to track the number of objects they had seen, to infer the existence of unobserved objects, and to learn and reason about categories.

Although the tasks in our experiments were all relatively simple, future work can apply our approach to more realistic settings. For example, a straightforward extension of our model can handle problems where objects vary along multiple perceptual dimensions and where observations are corrupted by perceptual noise. Discovery and identification problems may take several different forms, but probabilistic inference can help to explain how all of these problems are solved.

**Acknowledgments** We thank Bobby Han, Faye Han and Maureen Satyshur for running the experiments.

# References

[1] E. A. Tibbetts and J. Dale. Individual recognition: it is good to be different. *Trends in Ecology and Evolution*, 22(10):529–237, 2007.

[2] W. James. *Principles of psychology*. Holt, New York, 1890.

[3] R. M. Nosofsky. Attention, similarity, and the identification-categorization relationship. *Journal of Experimental Psychology: General*, 115:39–57, 1986.

[4] F. Xu and S. Carey. Infants' metaphysics: the case of numerical identity. *Cognitive Psychology*, 30:111–153, 1996.

[5] L. W. Barsalou, J. Huttenlocher, and K. Lamberts. Basing categorization on individuals and events. *Cognitive Psychology*, 36:203–272, 1998.

[6] L. J. Rips, S. Blok, and G. Newman. Tracing the identity of objects. *Psychological Review*, 113(1):1–30, 2006.

[7] A. McCallum and B. Wellner. Conditional models of identity uncertainty with application to noun coreference. In L. K. Saul, Y. Weiss, and L. Bottou, editors, *Advances in Neural Information Processing Systems 17*, pages 905–912. MIT Press, Cambridge, MA, 2005.

[8] B. Milch, B. Marthi, S. Russell, D. Sontag, D. L. Ong, and A. Kolobov. BLOG: Probabilistic models with unknown objects. In *Proceedings of the 19th International Joint Conference on Artificial Intelligence*, pages 1352–1359, 2005.

[9] J. Bunge and M. Fitzpatrick. Estimating the number of species: a review. *Journal of the American Statistical Association*, 88(421):364–373, 1993.

[10] R. G. Millikan. *On clear and confused ideas: an essay about substance concepts*. Cambridge University Press, New York, 2000.

[11] R. N. Shepard. Stimulus and response generalization: a stochastic model relating generalization to distance in psychological space. *Psychometrika*, 22:325–345, 1957.

[12] A. M. Leslie, F. Xu, P. D. Tremoulet, and B. J. Scholl. Indexing and the object concept: developing 'what' and 'where' systems. *Trends in Cognitive Science*, 2(1):10–18, 1998.

[13] J. D. Nichols. Capture-recapture models. *Bioscience*, 42(2):94–102, 1992.

[14] G. Csibra and A. Volein. Infants can infer the presence of hidden objects from referential gaze information. *British Journal of Developmental Psychology*, 26:1–11, 2008.

[15] H. Jeffreys. *Theory of Probability*. Oxford University Press, Oxford, 1961.

[16] J. R. Anderson. The adaptive nature of human categorization. *Psychological Review*, 98(3): 409–429, 1991.

[17] J. Pitman. Combinatorial stochastic processes, 2002. Notes for Saint Flour Summer School.

